# Connectionist Architectures for Multi-Speaker Phoneme Recognition

**John B. Hampshire II** and **Alex Waibel**
School of Computer Science
Carnegie Mellon University
Pittsburgh, PA 15213-3890

## ABSTRACT

We present a number of Time-Delay Neural Network (TDNN) based architectures for multi-speaker phoneme recognition (/b,d,g/ task). We use speech of two females and four males to compare the performance of the various architectures against a baseline recognition rate of 95.9% for a single TDNN on the six-speaker /b,d,g/ task. This series of modular designs leads to a highly modular multi-network architecture capable of performing the six-speaker recognition task at the speaker *de*pendent rate of 98.4%. In addition to its high recognition rate, the so-called "Meta-Pi" architecture learns — without direct supervision — to recognize the speech of one particular male speaker using internal models of *other* male speakers exclusively.

## 1  INTRODUCTION

References [1,2] have show the Time-Delay Neural Network to be an effective classifier of acoustic phonetic speech from individual speakers. The objective of this research has been to extend the TDNN paradigm to the multi-speaker phoneme recognition task, with the eventual goal of producing connectionist structures capable of speaker *independent* phoneme recognition. In making the transition from single to multi-speaker tasks, we have focused on modular architectures that perform the over-all recognition task by integrating a number of smaller task-specific networks.

**Table 1:** A synopsis of multi-speaker /b,d,g/ recognition results for six TDNN-based architectures.

| Architecture | Type | Features | Size (connections) | Recognition Rate | |
|---|---|---|---|---|---|
| | | | | 3-speakers | 6-speakers |
| TDNN | baseline single net | | 6,233 | 97.3% | 95.9% |
| FSTDNN | single net | •Frequency shift invariance | (1-ply) 5,357 (2-ply) 6,947 | 96.8% 97.2% | — — |
| Multiple TDNNs | multi net | •arbitrated classification | 18,700 | 98.6% | 97.1 % |
| Modular TDNN | multi net | •2-stage training | 18,650 37,400 | 97.3% — | — 96.3% |
| SID | multi net | •2-stage training •Multiple TDNN modules | 144,000 | — | 98.3% |
| Meta-Pi | multi net | •2-stage training •Multiple TDNN modules •Bayesian MAP learning •no explicit speaker I.D. | 144,000 | — | 98.4% |

## 1.1  DATA

The experimental conditions for this research are detailed in [1]. Japanese speech data from six professional announcers (2 female, 4 male) was sampled for the /b, d, g/ phonemes (approximately 250 training and 250 testing tokens per phoneme, per speaker). Training for all of the modular architectures followed a general two-stage process: in the first stage, speaker-dependent modules were trained on speech tokens from specific individuals; in the second stage, the over-all modular structure was trained with speech tokens from all speakers.

## 1.2  RESULTS

Owing to the number of architectures investigated, we present only brief descriptions of each structure. Additional references are provided for readers interested in more detailed descriptions of particular architectures. Table 1 summarizes our recognition results for all of the network architectures described below. We list the type of architecture (single or multi network), the important features of the design, its over-all size (in terms of total connections), and its recognition performance on the specified multi-speaker task. There are two principal multi-speaker tasks: a three male task, and a four male/two female task: the six speaker task is considerably more difficult than its three speaker counterpart, owing to the higher acoustic variance of combined male/female speech.

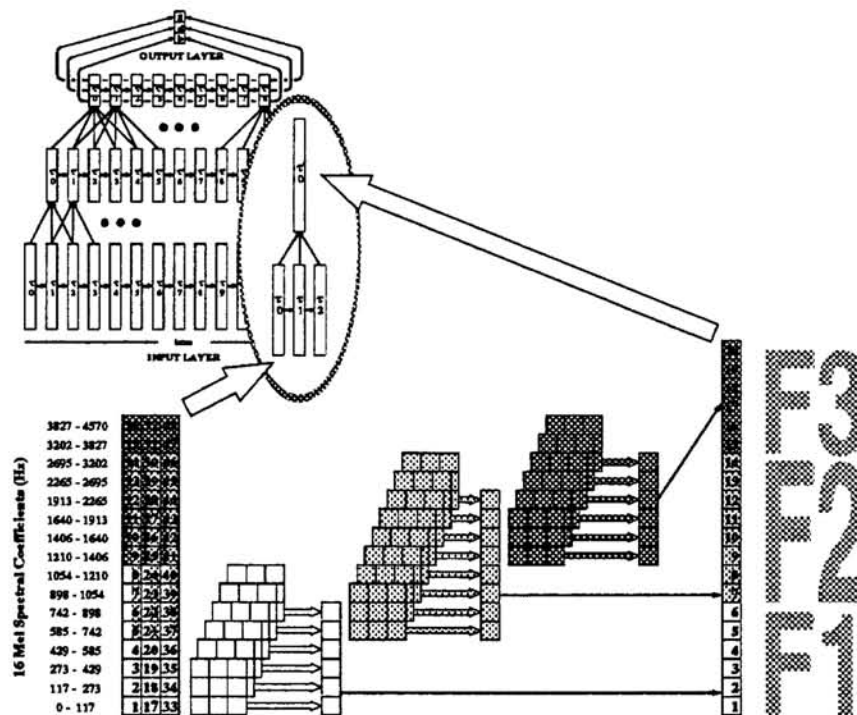

**Figure 1:** The Frequency Shifting TDNN (FSTDNN) architecture.

## 2   ARCHITECTURE DESCRIPTIONS

**TDNN:** The TDNN [1,2] serves as our baseline multi-speaker experiment. Its recognition performance on single speaker speech is typically 98.5% [1,3]. The high acoustic variance of speech drawn from six speakers — two of whom are female — reduces the TDNN's performance significantly (95.9%). This indicates that architectures capable of adjusting to markedly different speakers are necessary for robust multi-speaker and speaker-independent recognition.

**FSTDNN:** In this design, a frequency shift invariant feature is added to the original TDNN paradigm. The resulting architecture maps input speech into a first hidden layer with three frequency ranges roughly corresponding to the three formants F1 – F3 (see figure 1). Two variations of the basic design have been tested [4]: the first is a "one-ply" architecture (depicted in the figure), while the second is a "two-ply" structure that uses two plies of input to first hidden layer connections. While the frequency shift invariance of this architecture has intuitive appeal, the resulting network has a very small number of unique connections from the input to the first hidden layer (~ 30, 1-ply). This paucity of connections has two ramifications. First, it creates a crude replica of the input layer state in the first hidden layer; as a result, feature detectors that form in the connections between the input and first hidden layers of the standard TDNN are now formed in the connections between the first and second hidden layers of the FSTDNN. Second, the crude input to first hidden layer replication results in some loss of information; thus, the feature detectors of the FSTDNN operate on what can be viewed as a degraded version of

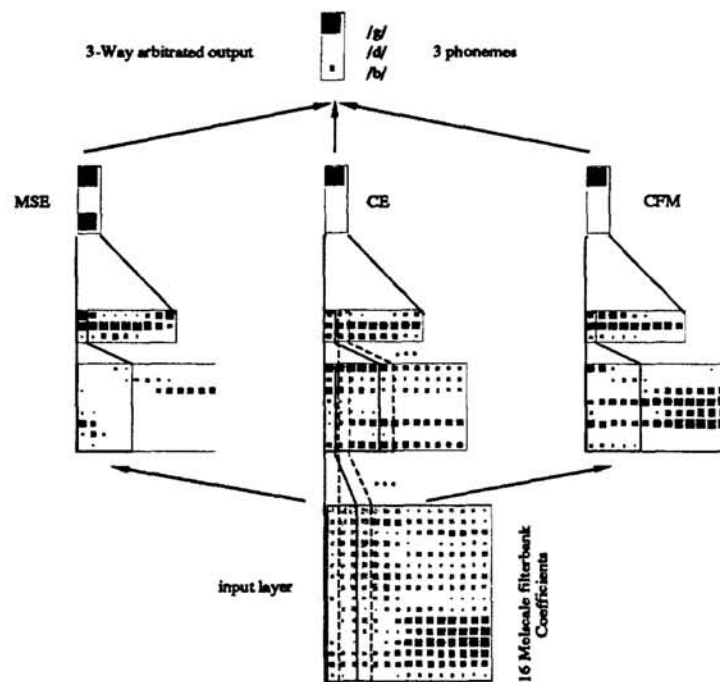

**Figure 2:** The Multiple TDNN architecture: three identical networks trained with three different objective functions.

the original input. The resulting over-all structure's recognition performance is typically worse (∼ 97%) than the baseline TDNN for the multi-speaker /b,d,g/ task.

**Multiple TDNN:** This design employs three TDNNs trained with the MSE, Cross Entropy [5], and CFM [3] objective functions (see figure 2). The different objective functions used to train the TDNNs form consistently different internal representations of the speech signal. We exploit these differing representations by using the (potentially) conflicting outputs of the three networks to form a global arbitrated classification decision. Taking the normalized sum of the three networks' outputs constitutes a simple arbitration scheme that typically reduces the single TDNN error rate by 30%.

**Modular TDNN:** In this design, we use the connection strengths of TDNNs fully trained on individual speakers to form the initial connections of a larger multi-speaker network. This resulting network's higher layer connections are retrained [6] to produce the final multi-speaker network. This technique allows us to integrate speaker-dependent networks into a larger structure, limiting the over-all training time and network complexity of the final multi-speaker architecture. The 3-speaker modular TDNN (shown in figures 3 and 4) shows selective response to different tokens of speech. In figure 3, the network responds to a /d/ phone with only one sub-network (associated with speaker "MNM"). In fact, this /d/ *is* spoken by "MNM". In figure 4, the same network responds to a /b/ phone spoken by "MHT" with all sub-networks. This selective response to utterances indicates that the network is sensitive to utterances that are prototypical for all speakers as well

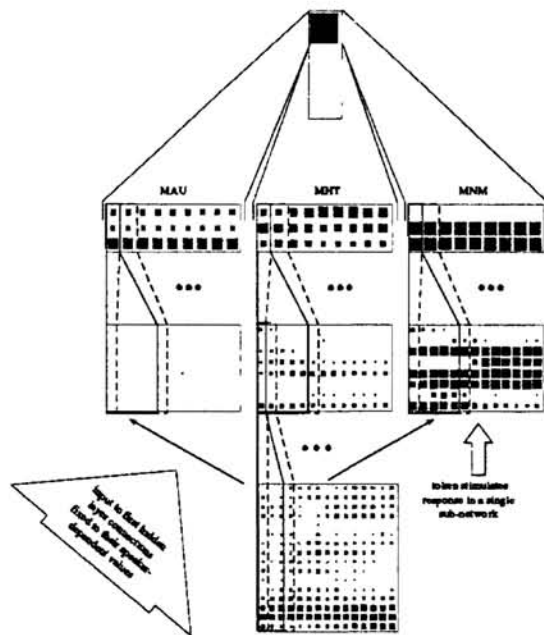

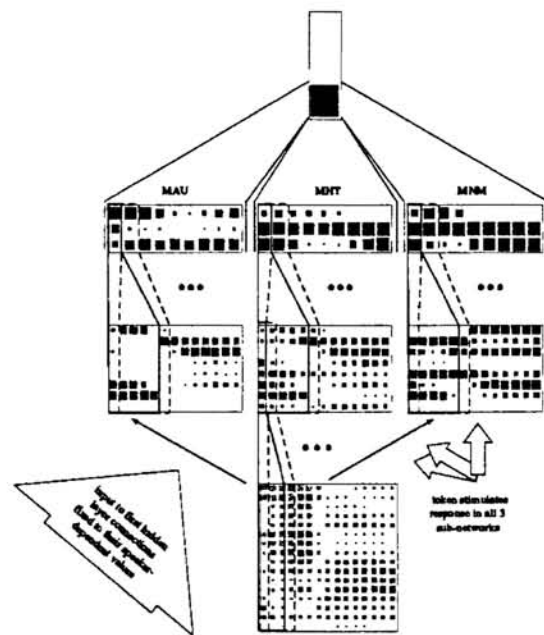

**Figure 3:** 3-speaker Modular TDNN responding to input with one module.

**Figure 4:** 3-speaker Modular TDNN responding to input with three modules.

as those that are unique to an individual. The recognition rate for the 3-speaker modular TDNN is comparable to the baseline TDNN rate (97.3%); however, the 6-speaker modular TDNN (not shown) yields a substantially lower recognition rate (96.3%). We attribute this degraded performance to the manner in which this modular structure integrates its sub-networks. In particular, the sub-networks are integrated by the connections from the second hidden to output layers. This scheme uses a very small number of connections to perform the integrating function. As the number of speakers increases and the acoustic variance of their speech becomes significant, the connection topology becomes inadequate for the increasingly complex integration function. Interconnecting the sub-networks between the first and second hidden layers would probably improve performance, but the improvement would be at the expense of modularity. We tried using a "Connectionist Glue" enhancement to the 6-speaker network [4], but found that it did not result in a significant recognition improvement.

**Stimulus Identification (SID) network:** This network architecture is conceptually very similar to the Integrated Neural Network (INN) [7]. Figure 5 illustrates the network in block diagram form. Stimulus specific networks (in this case, multiple TDNNs) are trained to recognize the speech of an individual. Each of these multiple TDNNs forms a module in the over-all network. The modules are integrated by a superstructure (itself a multiple TDNN) trained to recognize the identity of the input stimulus (speaker). The output activations of the integrating superstructure constitute multiplicative connections that gate the outputs of the modules in order to form a global classification decision.

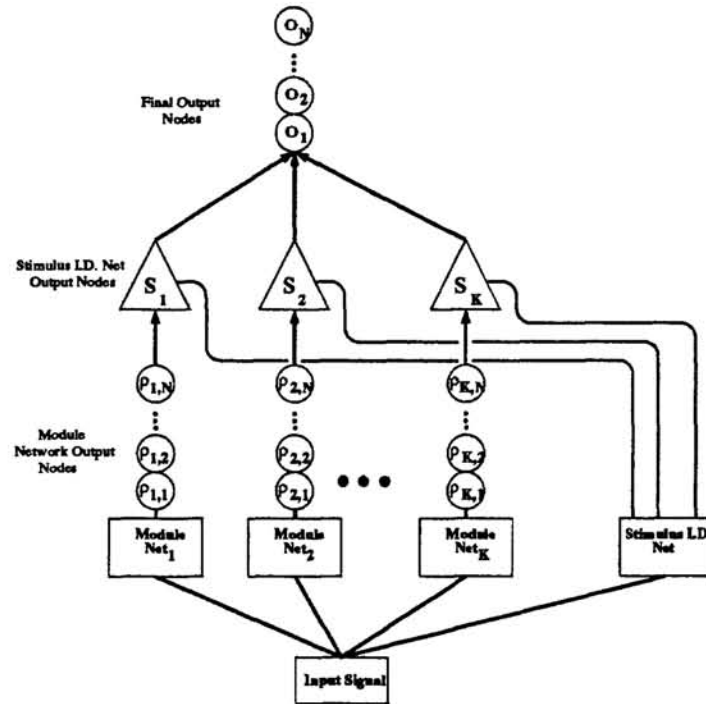

**Figure 5:** A block diagram of the Stimulus identification (SID) network, which is very similar to the Integrated Neural Network (INN) [7].

Reference [8] details the SID network's performance. The major advantages of this architecture are its high degree of modularity (all modules and the integrating superstructure can be trained independently) and it's high recognition rate (98.3%). It's major disadvantage is that it has no explicit mechanism for handling new speakers (see [8]).

**The Meta-Pi Network:** This network architecture is very similar to the SID network. Figure 6 illustrates the network in action. Stimulus specific networks (in this case, multiple TDNNs) are trained to recognize the speech of an individual. Each of these multiple TDNNs forms a module in the over-all network. The modules are integrated by a superstructure (itself a multiple TDNN) trained in Bayesian MAP fashion to maximize the phoneme recognition rate of the over-all structure: the equations governing the error back-propagation through the Meta-Pi superstructure link the global objective function with the output states of the network's speaker-dependent modules [8]. As with the the SID network, the output activations of the integrating superstructure constitute multiplicative connections that gate the outputs of the modules in order to form a global classification decision. However, as mentioned above, the integrating superstructure is *not* trained independently from the modules it integrates. While this Bayesian MAP training procedure is not as modularized as the SID network's training procedure, the resulting recognition rate is comparable. Additionally, the Meta-Pi network forms very broad representations of speaker *types* in order to perform its integration task. Reference [8] shows that *the Meta-Pi superstructure learns — without direct supervision — to perform its integra-*

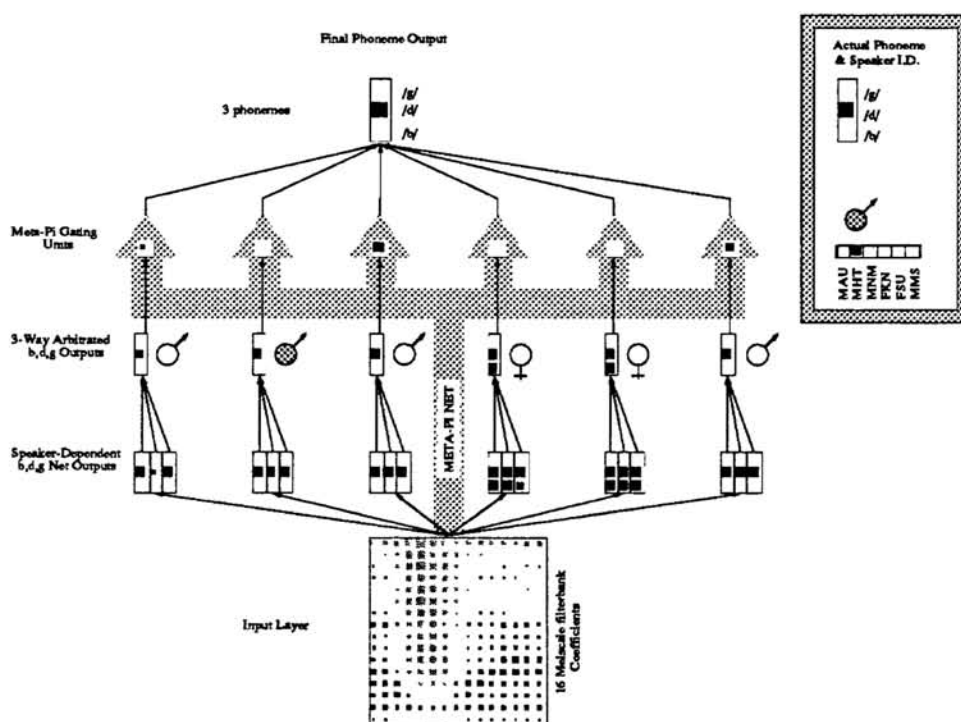

**Figure 6:** The Meta-Pi network responding to the speech of one male (MHT) using models of *other* males' speech exclusively.

*tion function based on gross formant features of the speakers being processed*; explicit speaker identity is irrelevant. A by-product of this learning procedure and the general representations that it forms is that the Meta-Pi network *learns to recognize the speech of one male using modules trained for* other *males exclusively* (see figure 6 and [8]).

## 3   CONCLUSION

We have presented a number of TDNN-based connectionist architectures for multi-speaker phoneme recognition. The Meta-Pi network combines the best features of a number of these designs with a Bayesian MAP learning rule to form a connectionist classifier that performs multi-speaker phoneme recognition at speaker-*de*pendent rates. We believe that the Meta-Pi network's ability to recognize the speech of one male using only models of other male speakers is significant. It suggests speech recognition systems that can maintain their own database of speaker models, adapting to new speakers when possible, spawning new speaker-dependent learning processes when necessary, and eliminating redundant or obsolete speaker-dependent modules when appropriate. The one major disadvantage of the Meta-Pi network is its size. We are presently attempting to reduce the network's size by 67% (target size: 48,000 connections) without a statistically significant loss in recognition performance.

**Acknowledgements**

We wish to thank Bell Communications Research, ATR Interpreting Telephony Research Laboratories, and the National Science Foundation (EET-8716324) for their support of this research. We thank Bellcore's David Burr, Daniel Kahn, and Candace Kamm and Seimens' Stephen Hanson for their comments and suggestions, all of which served to improve this work. We also thank CMU's Warp/iWarp[1] group for their support of our computational requirements. Finally, we thank Barak Pearlmutter, Dean Pomerleau, and Roni Rosenfeld for their stimulating conversations, insight, and constructive criticism.

## Footnotes

[1]iWarp is a registered trademark of Intel Corporation.

# References

[1] Waibel, A., Hanazawa, T., Hinton, G., Shikano, K., and Lang, K., "Phoneme Recognition Using Time-Delay Neural Networks," *IEEE Transactions on Acoustics, Speech and Signal Processing*, vol. ASSP-37, March, 1989, pp. 328-339.

[2] Lang, K. "A Time-Delay Neural Network Architecture for Speech Recognition," Ph.D. Dissertation, *Carnegie Mellon University technical report CMU-CS-89-185*, July, 31, 1989.

[3] Hampshire, J., Waibel, A., "A Novel Objective Function for Improved Phoneme Recognition Using Time-Delay Neural Networks," *Carnegie Mellon University Technical Report CMU-CS-89-118*, March, 1989. *A shorter version of this technical report is published in the IEEE Proceedings of the 1989 International Joint Conference on Neural Networks, vol. 1, pp. 235-241.*

[4] Hampshire, J., Waibel, A., "Connectionist Architectures for Multi-Speaker Phoneme Recognition," *Carnegie Mellon University Technical Report CMU-CS-89-167*, August, 1989.

[5] Hinton, G. E., "Connectionist Learning Procedures," *Carnegie Mellon University Technical Report CMU-CS-87-115 (version 2)*, December, 1987, pg. 14.

[6] Waibel, A., Sawai, H., and Shikano, K., "Modularity and Scaling in Large Phonemic Neural Networks", *IEEE Transactions on Acoustics, Speech and Signal Processing*, vol. ASSP-37, December, 1989, pp. 1888-1898.

[7] Matsuoka, T., Hamada, H., and Nakatsu, R., "Syllable Recognition Using Integrated Neural Networks," *IEEE Proceedings of the 1989 International Joint Conference on Neural Networks*, Washington, D.C., June 18-22, 1989, vol. 1, pp. 251-258.

[8] Hampshire, J., Waibel, A., "The Meta-Pi Network: Building Distributed Knowledge Representations for Robust Pattern Recognition," *Carnegie Mellon University Technical Report CMU-CS-89-166*, August, 1989.

